# Stochastic Hillclimbing as a Baseline Method for Evaluating Genetic Algorithms

**Ari Juels**
Department of Computer Science
University of California at Berkeley*

**Martin Wattenberg**
Department of Mathematics
University of California at Berkeley†

## Abstract

We investigate the effectiveness of stochastic hillclimbing as a baseline for evaluating the performance of genetic algorithms (GAs) as combinatorial function optimizers. In particular, we address two problems to which GAs have been applied in the literature: Koza's 11-multiplexer problem and the jobshop problem. We demonstrate that simple stochastic hillclimbing methods are able to achieve results comparable or superior to those obtained by the GAs designed to address these two problems. We further illustrate, in the case of the jobshop problem, how insights obtained in the formulation of a stochastic hillclimbing algorithm can lead to improvements in the encoding used by a GA.

## 1   Introduction

*Genetic algorithms* (GAs) are a class of randomized optimization heuristics based loosely on the biological paradigm of natural selection. Among other proposed applications, they have been widely advocated in recent years as a general method for obtaining approximate solutions to hard combinatorial optimization problems using a minimum of information about the mathematical structure of these problems. By means of a general "evolutionary" strategy, GAs aim to maximize an objective or *fitness* function $f : S \to \mathbf{R}$ over a combinatorial space $S$, i.e., to find some state $s \in S$ for which $f(s)$ is as large as possible. (The case in which $f$ is to be minimized is clearly symmetrical.) For a detailed description of the algorithm see, for example, [7], which constitutes a standard text on the subject.

In this paper, we investigate the effectiveness of the GA in comparison with that of stochastic hillclimbing (SH), a probabilistic variant of hillclimbing. As the term

---
*Supported in part by NSF Grant CCR-9505448. E-mail: **juels@cs.berkeley.edu**
†E-mail: **wattenbe@math.berkeley.edu**

"hillclimbing" suggests, if we view an optimization problem as a "landscape" in which each point corresponds to a solution $s$ and the "height" of the point corresponds to the fitness of the solution, $f(s)$, then hillclimbing aims to ascend to a peak by repeatedly moving to an adjacent state with a higher fitness.

A number of researchers in the GA community have already addressed the issue of how various versions of hillclimbing on the space of bitstrings, $\{0, 1\}^n$, compare with GAs [1] [4] [9] [18] [15]. Our investigations in this paper differ in two important respects from these previous ones. First, we address more sophisticated problems than the majority of these studies, which make use of test functions developed for the purpose of exploring certain landscape characteristics. Second, we consider hillclimbing algorithms based on operators in some way "natural" to the combinatorial structures of the problems to which we are seeking solutions, very much as GA designers attempt to do. In one of the two problems in this paper, our SH algorithm employs an encoding exactly identical to that in the proposed GA. Consequently, the hillclimbing algorithms we consider operate on structures other than bitstrings.

Constraints in space have required the omission of a great deal of material found in the full version of this paper. This material includes the treatment of two additional problems: the NP-complete Maximum Cut Problem [11] and an NP-complete problem known as the multiprocessor document allocation problem (MDAP). Also in the full version of this paper is a substantially more thorough exposition of the material presented here. The reader is encouraged to refer to [10], available on the World Wide Web at http://www.cs.berkeley.edu/~juels/.

## 2   Stochastic Hillclimbing

The SH algorithm employed in this paper searches a discrete space $S$ with the aim of finding a state whose fitness is as high (or as low) as possible. The algorithm does this by making successive improvements to some current state $\sigma \in S$. As is the case with genetic algorithms, the form of the states in $S$ depends upon how the designer of the SH algorithm chooses to encode the solutions to the problems to be solved: as bitstrings, permutations, or in some other form. The local improvements effected by the SH algorithm are determined by the *neighborhood structure* and the fitness function $f$ imposed on $S$ in the design of the algorithm. We can consider the neighborhood structure as an undirected graph $G$ on vertex set $S$. The algorithm attempts to improve its current state $\sigma$ by making a transition to one of the neighbors of $\sigma$ in $G$. In particular, the algorithm chooses a state $\tau$ according to some suitable probability distribution on the neighbors of $\sigma$. If the fitness of $\tau$ is as least as good as that of $\sigma$ then $\tau$ becomes the new current state, otherwise $\sigma$ is retained. This process is then repeated

## 3   GP and Jobshop

### 3.1   The Experiments

In this section, we compare the performance of SH algorithms with that of GAs proposed for two problems: the jobshop problem and Koza's 11-multiplexer problem. We gauge the performance of the GA and SH algorithms according to the fitness of the best solution achieved after a fixed number of function evaluations, rather than the running time of the algorithms. This is because evaluation of the fitness function generally constitutes the most substantial portion of the execution time of the optimization algorithm, and accords with standard practice in the GA community.

## 3.2  Genetic Programming

"Genetic programming" (GP) is a method of enabling a genetic algorithm to search a potentially infinite space of computer programs, rather than a space of fixed-length solutions to a combinatorial optimization problem. These programs take the form of Lisp symbolic expressions, called *S-expressions*. The S-expressions in GP correspond to programs which a user seeks to adapt to perform some pre-specified task. Details on GP, an increasingly common GA application, and on the 11-multiplexer problem which we address in this section, may be found, for example, in [13] [12] [14].

The boolean 11-multiplexer problem entails the generation of a program to perform the following task. A set of 11 distinct inputs is provided, with labels $a_0, a_1, a_2, d_0, d_1, \ldots, d_7$, where $a$ stands for "address" and $d$ for "data". Each input takes the value 0 or 1. The task is to output the value $d_m$, where $m = a_0 + 2a_1 + 4a_2$. In other words, for any 11-bit string, the input to the "address" variables is to be interpreted as an index to a specific "data" variable, which the program then yields as output. For example, on input $a_1 = 1, a_0 = a_2 = 0$, and $d_2 = 1, d_0 = d_1 = d_3 = \ldots = d_7 = 0$, a correct program will output a '1', since the input to the 'a' variables specifies address 2, and variable $d_2$ is given input 1.

**The GA**  Koza's GP involves the use of a GA to generate an S-expression corresponding to a correct 11-multiplexer program. An S-expression comprises a tree of LISP *operators* and *operands*, operands being the set of data to be processed — the leaves of the tree — and operators being the functions applied to these data and internally in the tree. The nature of the operators and operands will depend on the problem at hand, since different problems will involve different sets of inputs and will require different functions to be applied to these inputs. For the 11-multiplexer problem in particular, where the goal is to create a specific boolean function, the operands are the input bits $a_0, a_1, a_2, d_0, d_1, \ldots, d_7$, and the operators are AND, OR, NOT, and IF. These operators behave as expected: the subtree (AND $a_1$ $a_2$), for instance, yields the value $a_1 \wedge a_2$. The subtree (IF $a_1$ $d_4$ $d_3$) yields the value $d_4$ if $a_1 = 0$ and $d_3$ if $a_1 = 1$ (and thus can be regarded as a "3-multiplexer"). NOT and OR work similarly. An S-expression constitutes a tree of such operators, with operands at the leaves. Given an assignment to the operands, this tree is evaluated from bottom to top in the obvious way, yielding a 0 or 1 output at the root.

Koza makes use of a "mating" operation in his GA which swaps subexpressions between two such S-expressions. The subexpressions to be swapped are chosen uniformly at random from the set of all subexpressions in the tree. For details on selection in this GA, see [13]. The fitness of an S-expression is computed by evaluating it on all 2048 possible inputs, and counting the number of correct outputs. Koza does not employ a mutation operator in his GA.

**The SH Algorithm**  For this problem, the initial state in the SH algorithm is an S-expression consisting of a single operand chosen uniformly at random from $\{a_0, a_1, a_2, d_0, \ldots, d_7\}$. A transition in the search space involves the random replacement of an arbitrary node in the S-expression. In particular, to select a neighboring state, we chose a node uniformly at random from the current tree and replace it with a node selected randomly from the set of all possible operands and operators. With probability $\frac{1}{2}$ the replacement node is drawn uniformly at random from the set of operands $\{a_0, a_1, a_2, d_0, \ldots, d_7\}$, otherwise it is drawn uniformly at random from the set of operators, $\{$AND, OR, NOT, IF$\}$. In modifying the nodes of the S-expression in this way, we may change the number of inputs they require. By changing an AND node to a NOT node, for instance, we reduce the number of inputs taken by the node from 2 to 1. In order to accommodate such changes, we do

the following. Where a replacement reduces the number of inputs taken by a node, we remove the required number of children from that node uniformly at random. Where, on the other hand, a replacement increases the number of inputs taken by a node, we add the required number of children chosen uniformly at random from the set of operands $\{a_0, a_1, a_2, d_0, \ldots, d_7\}$. A similar, though somewhat more involved approach of this kind, with additional experimentation using simulated annealing, may be found in [17].

**Experimental Results**   In the implementation described in [14], Koza performs experiments with a GA on a pool of 4000 expressions. He records the results of 54 runs. These results are listed in the table below. The average number of function evaluations required to obtain a correct program is not given in [14]. In [12], however, where Koza performs a series of 21 runs with a slightly different selection scheme, he finds that the average number of function evaluations required to find a correct S-expression is 46,667.

In 100 runs of the SH algorithm, we found that the average time required to obtain a correct S-expression was 19,234.90 function evaluations, with a standard deviation of 5179.45. The minimum time to find a correct expression in these runs was 3733, and the maximum, 73,651. The average number of nodes in the correct S-expression found by the SH algorithm was 88.14; the low was 42, the high, 242, and the standard deviation, 29.16.

The following table compares the results presented in [14], indicated by the heading "GP", with those obtained using stochastic hillclimbing, indicated by "SH". We give the fraction of runs in which a correct program was found after a given number of function evaluations. (As this fraction was not provided for the 20000 iteration mark in [14], we omit the corresponding entry.)

| *Functionevaluations* | GP | SH |
|:---:|:---:|:---:|
| 20000 | | 61 % |
| 40000 | 28 % | 98 % |
| 60000 | 78 % | 99 % |
| 80000 | 90 % | 100 % |

We observe that the performance of the SH is substantially better than that of the GA. It is interesting to note – perhaps partly in explanation of the SH algorithm's success on this problem – that the SH algorithm formulated here defines a neighborhood structure in which there are *no strict local minima*. Remarkably, this is true for any boolean formula. For details, as well as an elementary proof, see the full version of this paper [10].

## 3.3   Jobshop

Jobshop is a notoriously difficult NP-complete problem [6] that is hard to solve even for small instances. In this problem, a collection of $J$ jobs are to be scheduled on $M$ machines (or processors), each of which can process only one task at a time. Each job is a list of $M$ tasks which must be performed in order. Each task must be performed on a specific machine, and no two tasks in a given job are assigned to the same machine. Every task has a fixed (integer) processing time. The problem is to schedule the jobs on the machines so that all jobs are completed in the shortest overall time. This time is referred to as the *makespan*.

Three instances formulated in [16] constitute a standard benchmark for this problem: a 6 job, 6 machine instance, a 10 job, 10 machine instance, and a 20 job, 5

machine instance. The 6x6 instance is now known to have an optimal makespan of 55. This is very easy to achieve. While the optimum value for the 10x10 problem is known to be 930, this is a difficult problem which remained unsolved for over 20 years [2]. A great deal of research has also been invested in the similarly challenging 20x5 problem, for which an optimal value of 1165 has been achieved, and a lower bound of 1164 [3].

A number of papers have considered the application of GAs to scheduling problems. We compare our results with those obtained in Fang et al. [5], one of the more recent of these articles.

**The GA**   Fang et al. encode a jobshop schedule in the form of a string of integers, to which their GA applies a conventional crossover operator. This string contains $JM$ integers $a_1, a_2, \ldots, a_{JM}$ in the range $1..J$. A circular list $C$ of jobs, initialized to $(1, 2, \ldots, J)$ is maintained. For $i = 1, 2, \ldots, JM$, the first uncompleted task in the $(a_i \bmod |C|)^{th}$ job in $C$ is scheduled in the earliest plausible timeslot. A *plausible* timeslot is one which comes after the last scheduled task in the current job, and which is at least as long as the processing time of the task to be scheduled. When a job is complete, it is removed from $C$. Fang et al. also develop a highly specialized GA for this problem in which they use a scheme of increasing mutation rates and a technique known as GVOT (Gene-Variance based Operator Targeting). For the details see [5].

**The SH Algorithm**   In our SH algorithm for this problem, a schedule is encoded in the form of an ordering $\sigma_1, \sigma_2, \ldots, \sigma_{JM}$ of $JM$ markers. These markers have colors associated with them: there are exactly $M$ markers of each color of $1, \ldots, J$. To construct a schedule, $\sigma$ is read from left to right. Whenever a marker with color $k$ is encountered, the next uncompleted task in job $k$ is scheduled in the earliest plausible timeslot. Since there are exactly $M$ markers of each color, and since every job contains exactly $M$ tasks, this decoding of $\sigma$ yields a complete schedule. Observe that since markers of the same color are interchangeable, many different ordering $\sigma$ will correspond to the same scheduling of tasks.

To generate a neighbor of $\sigma$ in this algorithm, a marker $\sigma_i$ is selected uniformly at random and moved to a new position $j$ chosen uniformly at random. To achieve this, it is necessary to shift the subsequence of markers between $\sigma_i$ and $\sigma_j$ (including $\sigma_j$) one position in the appropriate direction. If $i < j$, then $\sigma_{i+1}, \sigma_{i+2}, \ldots, \sigma_j$ are shifted one position to the left in $\sigma$. If $i > j$, then $\sigma_j, \sigma_{j+1}, \ldots, \sigma_{i-1}$ are shifted one position to the right. (If $i = j$, then the generated neighbor is of course identical to $\sigma$.) For an example, see the full version of this paper [10].

Fang et al. consider the makespan achieved after 300 iterations of their GVOT-based GA on a population of size 500. We compare this with an SH for which each experiment involves 150,000 iterations. In both cases therefore, a single execution of the algorithm involves a total of 150,000 function evaluations. Fang et al. present their average results over 10 trials, but do not indicate how they obtain their "best". We present the statistics resulting from 100 executions of the SH algorithm.

| | 10x10 Jobshop | | 20x5 Jobshop | |
|---|---|---|---|---|
| | GA | SH | GA | SH |
| Mean | 977 | 966.96 | 1215 | 1202.40 |
| SD | | 13.15 | | 12.92 |
| High | | 997 | | 1288 |
| Low | 949 | 938 | 1189 | 1173 |
| Best Known | | *930* | | *1165* |

As can be seen from the above table, the performance of the SH algorithm appears to be as good as or superior to that of the GA.

## 3.4    A New Jobshop GA

In this section, we reconsider the jobshop problem in an attempt to formulate a new GA encoding. We use the same encoding as in the SH algorithm described above: $\sigma$ is an ordering $\sigma_1, \sigma_2, \ldots, \sigma_{JM}$ of the $JM$ markers, which can be used to construct a schedule as before. We treated markers of the same color as effectively equivalent in the SH algorithm. Now, however, the label of a marker (a unique integer in $\{1, \ldots, JM\}$) will play a role.

The basic step in the crossover operator for this GA as applied to a pair $(\sigma, \tau)$ of orderings is as follows. A label $i$ is chosen uniformly at random from the set $\{1, 2, \ldots, JM\}$. In $\sigma$, the marker with label $i$ is moved to the position occupied by $i$ in $\tau$; conversely, the marker with label $i$ in $\tau$ is moved to the position occupied by that marker in $\sigma$. In both cases, the necessary shifting is performed as before. Hence the idea is to move a single marker in $\sigma$ (and in $\tau$) to a new position as in the SH algorithm; instead of moving the marker to a random position, though, we move it to the position occupied by that marker in $\tau$ (and $\sigma$, respectively). The full crossover operator picks two labels $j \leq k$ uniformly at random from $\{1, 2, \ldots, JM\}$, and performs this basic operation first for label $j$, then $j + 1$, and so forth, through $k$. The mutation operator in our GA performs exactly the same operation as that used to generate a neighbor in the SH algorithm. A marker $\sigma_i$ is chosen uniformly at random and moved to a new position $j$, chosen uniformly at random. The usual shifting operation is then performed. Observe how closely the crossover and mutation operators in this GA for the jobshop problem are based on those in the corresponding SH algorithm.

Our GA includes, in order, the following phases: evaluation, elitist replacement, selection, crossover, and mutation. In the evaluation phase, the fitnesses of all members of the population are computed. Elitist replacement substitutes the fittest permutation from the evaluation phase of the previous iteration for the least fit permutation in the current population (except, of course, in the first iteration, in which there is no replacement). Because of its simplicity and its effectiveness in practice, we chose to use binary stochastic tournament selection (see [8] for details). The crossover step in our GA selects $\frac{P}{2}$ pairs uniformly at random without replacement from the population and applies the mating operator to each of these pairs independently with probability 0.6. The number of mutations performed on a given permutation in a single iteration is binomial with parameter $p = \frac{1}{n}$. The population in our GA is initialized by selecting every individual uniformly at random from $S_n$.

We execute this GA for 300 iterations on a population of size 500. Results of 100 experiments performed with this GA are indicated in the following table by "new GA". For comparison, we again give the results obtained by the GA of Fang et al. and the SH algorithm described in this paper.

|  | 10x10 Jobshop | | | 20x5 Jobshop | | |
|---|---|---|---|---|---|---|
|  | new GA | GA | SH | new GA | GA | SH |
| Mean | 956.22 | 977 | 965.64 | 1193.21 | 1215 | 1204.89 |
| SD | 8.69 | | 10.56 | 7.38 | | 12.92 |
| High | 976 | | 996 | 1211 | | 1241 |
| Low | 937 | 949 | 949 | 1174 | 1189 | 1183 |
| Best Known | | *930* | | | *1165* | |

## 4  Conclusion

As black-box algorithms, GAs are principally of interest in solving problems whose combinatorial structure is not understood well enough for more direct, problem-specific techniques to be applied. As we have seen in regard to the two problems presented in this paper, stochastic hillclimbing can offer a useful gauge of the performance of the GA. In some cases it shows that a GA-based approach may not be competitive with simpler methods; at others it offers insight into possible design decisions for the GA such as the choice of encoding and the formulation of mating and mutation operators. In light of the results presented in this paper, we hope that designers of black-box algorithms will be encouraged to experiment with stochastic hillclimbing in the initial stages of the development of their algorithms.

## References

[1] D. Ackley. *A Connectionist Machine for Genetic Hillclimbing.* Kluwer Academic Publishers, 1987.

[2] D. Applegate and W. Cook. A computational study of the job-shop problem. *ORSA Journal of Computing,* 3(2), 1991.

[3] J. Carlier and E. Pinson. An algorithm for solving the jobshop problem. *Mngmnt. Sci.,* 35:(2):164–176, 1989.

[4] L. Davis. Bit-climbing, representational bias, and test suite design. In Belew and Booker, editors, *ICGA-4,* pages 18–23, 1991.

[5] H. Fang, P. Ross, and D. Corne. A promising GA approach to job-shop scheduling, rescheduling, and open-shop scheduling problems. In Forrest, editor, *ICGA-5,* 1993.

[6] M. Garey and D. Johnson. *Computers and Intractability.* W.H. Freeman and Co., 1979.

[7] D. Goldberg. *Genetic Algorithms in Search, Optimization, and Machine Learning.* Addison Wesley, 1989.

[8] D. Goldberg and K. Deb. A comparative analysis of selection schemes used in GAs. In *FOGA-2,* pages 69–93, 1991.

[9] K. De Jong. *An Analysis of the Behavior of a Class of Genetic Adaptive Systems.* PhD thesis, University of Michigan, 1975.

[10] A. Juels and M. Wattenberg. Stochastic hillclimbing as a baseline method for evaluating genetic algorithms. Technical Report CSD-94-834, UC Berkeley, CS Division, 1994.

[11] S. Khuri, T. Bäck, and J. Heitkötter. An evolutionary approach to combinatorial optimization problems. In *Procs. of CSC 1994,* 1994.

[12] J. Koza. *FOGA,* chapter A Hierarchical Approach to Learning the Boolean Multiplexer Function, pages 171–192. 1991.

[13] J. Koza. *Genetic Programming.* MIT Press, Cambridge, MA, 1991.

[14] J. Koza. The GP paradigm: Breeding computer programs. In Branko Souček and the IRIS Group, editors, *Dynamic, Genetic, and Chaotic Prog.,* pages 203–221. John Wiley and Sons, Inc., 1992.

[15] M. Mitchell, J. Holland, and S. Forrest. When will a GA outperform hill-climbing? In J.D. Cowen, G. Tesauro, and J. Alspector, editors, *Advances in Neural Inf. Processing Systems 6,* 1994.

[16] J. Muth and G. Thompson. *Industrial Scheduling.* Prentice Hall, 1963.

[17] U. O'Reilly and F. Oppacher. Program search with a hierarchical variable length representation: Genetic programing, simulated annealing and hill climbing. In *PPSN-3,* 1994.

[18] S. Wilson. GA-easy does not imply steepest-ascent optimizable. In Belew and Booker, editors, *ICGA-4,* pages 85–89, 1991.
